# An Analog Implementation of the Constant Statistics Constraint For Sensor Calibration

**John G. Harris and Yu-Ming Chiang**
Computational Neuro-Engineering Laboratory
Department of Computer and Electrical Engineering
University of Florida
Gainesville, FL 32611

## Abstract

We use the *constant statistics* constraint to calibrate an array of sensors that contains gain and offset variations. This algorithm has been mapped to analog hardware and designed and fabricated with a 2um CMOS technology. Measured results from the chip show that the system achieves invariance to gain and offset variations of the input signal.

## 1   Introduction

Transistor mismatches and parameter variations cause unavoidable nonuniformities from sensor to sensor. A one-time calibration procedure is normally used to counteract the effect of these fixed variations between components. Unfortunately, many of these variations fluctuate with time–either with operating point (such as data-dependent variations) or with external conditions (such as temperature). Calibrating these sensors one-time only at the "factory" is not suitable–much more frequent calibration is required. The sensor calibration problem becomes more challenging as an increasing number of different types of sensors are integrated onto VLSI chips at higher and higher integration densities. Ullman and Schechtman studied a simple gain adjustment algorithm but their method provides no mechanism for canceling additive offsets [10]. Scribner has addressed this nonuniformity correction problem in software using a neural network technique but it will be difficult to integrate this complex solution into analog hardware [9]. A number of researchers have studied sensors that output the time-derivative of the signal[9][4]. A simple time derivative

cancels any additive offset in the signal but also loses all of the DC and most of the low frequency temporal information present. The offset-correction method proposed by this paper, in effect, uses a time-derivative with an extremely long time constant thereby preserving much of the low-frequency information present in the signal. However, even if an ideal time-derivative approximation is used to cancel out additive offsets, the standard deviation process described in this paper can be used to factor out gain variations.

We hope to obtain some clues for sensory adaptation from neurobiological systems which possess a tremendous ability to adapt to the surrounding environment at multiple time-scales and at multiple stages of processing. Consider the following experiments:

- After staring at a single curved line ten minutes, human subjects report that the amount of curvature perceived appears to decrease. Immediately after training, the subjects then were shown a straight line and perceived it as slightly curved in the opposite direction[5].

- After staring long enough at an object in continuous motion, the motion seems to decrease with time. Immediately after adaptation, subjects perceive motion in the opposite direction when looking at stationary objects. This experiment is called the waterfall effect[2].

- Colors tend to look less saturated over time. Color after-images are perceived containing exactly the opponent colors of the original scene[1].

Though the purpose of these biological adaptation mechanisms is not clear, some theories suggest that these methods allow for fine-tuning the visual system through long-term averaging of measured visual parameters[10]. We will apply such continuous-calibration procedures to VLSI sensor calibration.

The real-world variable $x(t)$ is transduced by a nonlinear response curve into a measured variable $y(t)$. For a single operating point, the linear approximation can be written as:

$$y(t) = ax(t) + b \qquad (1)$$

with $a$ and $b$ being the multiplicative gain and additive offset respectively. The gain and offset values vary from pixel to pixel and may vary slowly over time. Current infra-red focal point arrays (IRFPAs) are limited by their inability to calibrate out component variations [3]. Typically, off-board digital calibration is used to correct nonuniformities in these detector arrays; Special calibration images are used to calibrate the system at startup. One-time calibration procedures such as these do not take into account other operating points and will fail to recalibrate for any drift in the parameters.

## 2   Implementing Natural Constraints

A continuous calibration system must take advantage of natural constraints available during the normal operation of the sensors. One theory holds that biological systems adapt to the long-term average of the stimulus. For example, the constraints for the three psychophysical examples mentioned above (curvature, motion and color adaptation) may rely on the following constraints:

- The average line is straight.

- The average motion is zero.

- The average color is gray.

The system adapts over time in the direction of this average, where the average must be taken over a very long time: from minutes to hours. We use two additional constraints for offset/gain normalization, namely:

- The average pixel intensities are identical.

- The variances of the input for each pixel are all identical.

Each of these constraints assumes that the photoarray is periodically moving in the real-world and that the average statistics each pixels sees should be constant when averaged over a very long time. In pathological situations where humans or machines are forced to stare at a single static scene for a long time, we violate this assumption.

We estimate the time-varying mean and variance by using an exponentially shaped window into the past. The equations for mean and variance are:

$$m(t) = \frac{1}{\tau} \int_0^\infty y(t - \Delta)e^{-\Delta/\tau} d\Delta \tag{2}$$

and

$$s(t) = \frac{1}{\tau} \int_0^\infty |y(t - \Delta) - m(t - \Delta)|e^{-\Delta/\tau} d\Delta \tag{3}$$

The $m(t)$ and $s(t)$ in Equation 2 and 3 can be expressed as low-pass filters with inputs $y(t)$ and $|y(t) - m(t)|$ respectively. To simplify the hardware implementation further, we chose the $L_1$ (absolute value) definition of variance instead of the more usual $L_2$ definition. The $L_1$ definition is an equally acceptable definition of signal variation in terms of the complete calibration system. Using this definition, no squares or square roots need be calculated. An added benefit of the $L_1$ norm is that it provides robustness to outliers in the estimation.

A zero-mean, unity variance[1] signal can then be produced with the following shift/normalization formula:

$$x(t) = \frac{y(t) - m(t)}{s(t)} \tag{4}$$

Equation 2, Equation 3 and Equation 4 constitute a new algorithm for continuously calibrating systems with gain and offset variations. Note that without additional apriori knowledge about the values of the gains and offsets, it is impossible to recover the true value of the signal $x(t)$ given an infinite history of $y(t)$. This is an ill-posed problem even with fixed but unknown gain and offset parameters for each sensor. All that can be done is to calibrate each sensor output to have zero offset and unity variance. After calibration, each sensor would therefore all have the same offset and variance when averaged over a long time. The fundamental assumption embedded

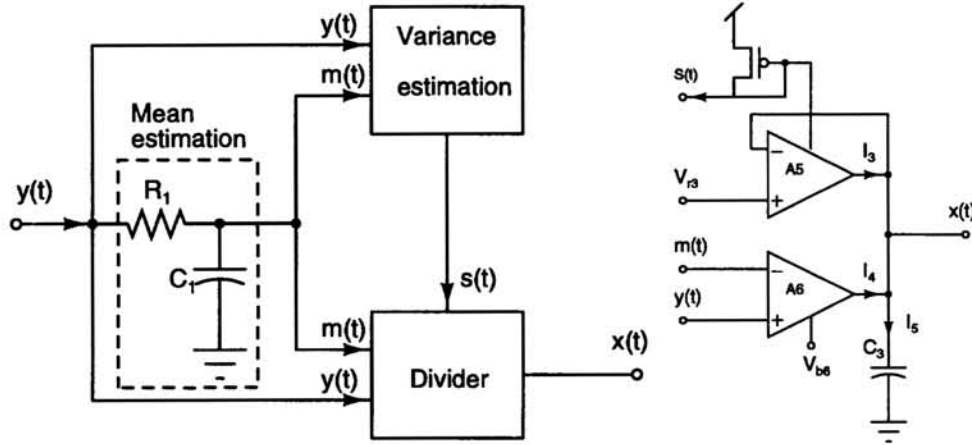

Figure 1: *Left: block diagram of continuous-time calibration system, Right: schematic of the divider circuit.*

in this algorithm is that each sensor measures real-world quantities with the same statistical properties (e.g., mean and variance). For example, this would mean that all pixels in a camera should eventually see the same average intensity when integrated over a long enough time. This assumption leads to other system-level constraints–in this case, the camera must be periodically moving.

We have successfully demonstrated this algorithm in software for the case of nonuniformity (gain and offset) correction of images [6]. Since there may be potentially thousands of sensors per chip, it is desirable to build calibration circuitry using subthreshold analog MOS technology to achieve ultra-low power consumption[8]. The next section describes the analog VLSI implementation of this algorithm.

## 3   Continuous-time calibration circuit

The block diagram of the continuous-time gain and offset calibration circuit is shown in Figure 1a. This system includes three building blocks: a mean estimation circuit, a variance estimation circuit and a divider circuit. As is shown, the mean of the signal can be easily extracted by a RC low-pass filter circuit. Since there may be potentially thousands of sensors per chip, it is desirable to build calibration circuitry using subthreshold analog MOS technology to achieve ultra-low power consumption[8].

Figure 2 shows the schematic of the variance estimation circuit. A full-wave rectifier [8] operating in the sub-threshold region is used to obtain the absolute value of the difference between the input and its mean. In the linear region, the current $I_{out}$ is proportional to $|y(t) - m(t)|$. As indicated in Equation 3, $I_{out}$ has to be low-pass filtered to obtain $s(t)$. In Figure 2, transconductance amplifiers $A_3$, $A_4$ and capacitor $C_2$ are used to form a current mode low-pass filter. For signals in the linear region, we can derive the Laplace transform of $V_1$ as:

$$V_1(s) = \frac{R}{RC_2 s + 1} I_{out}(s) \tag{5}$$

which is a first-order low-pass filter for $I_{out}$. The value of R is a function of several

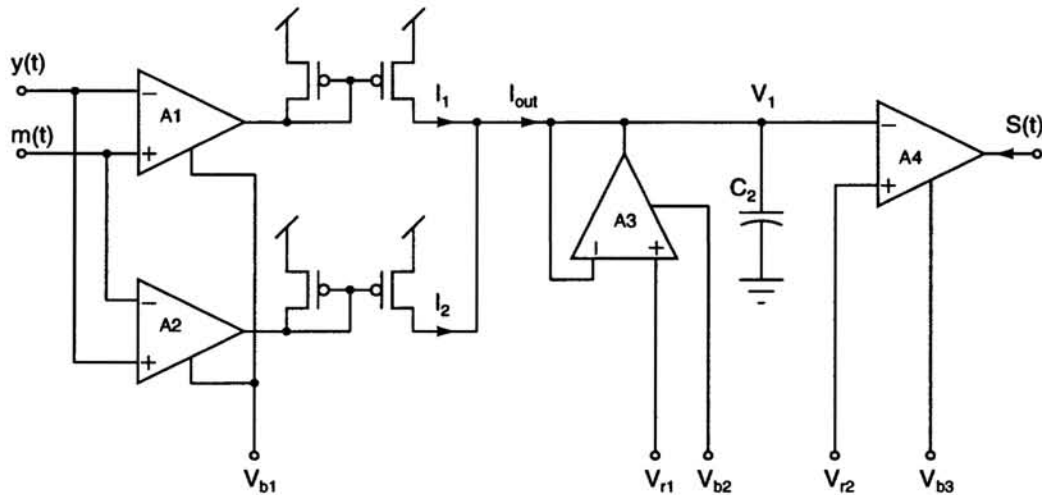

Figure 2: *Variance estimation circuit. The triangle symbols represent 5-transistor transconductance amplifiers that output a current proportional to the difference of their inputs.*

fabrication constants and an adjustable bias current. Figure 3(a) shows the expected linear relationship between the measured variance $s(t)$ and the peak-to-peak amplitude of the sine-wave input.

The third building block in the calibration system is the divider circuit shown in Figure 1b. A fed-back multiplier is used to enforce the constraint that $y(t) - m(t)$ is proportional to $x(t)s(t)$ which results in a scaled version of Equation 4. The characteristics of the divider have been measured and shown in Figure 3(b). With a fixed $V_{b6}$ and $m(t)$, we sweep $y(t)$ from $m(t) - 0.3V$ to $m(t) + 0.3V$ and measure the the change of output. A family of input/output characteristics with $s(t) =20$, 25, 30, 40, 50, 60 and 70nA is shown in Figure 3(b). The divider circuit has been tested up to frequencies of 45kHz.

The first version of the calibration circuit has been designed and fabricated in a 2-um CMOS technology. The chip includes the major parts of this calibration circuit: the variance estimation circuit and divider circuit. In our initial design, the mean estimation circuit, which is simply a RC low-pass filter, was built off-chip. However, it can be easily integrated on-chip using a transconductance amplifier and a capacitor.

The calibration results for a signal with gain and offset variations are shown in Figure 4. The input signal is a sine wave with a severe gain and offset jump as shown at the top of Figure 4. At the middle of Figure 4, the convergence of the variance estimation is illustrated. It takes a short time for the circuit to converge after any change of the mean or variance or of the input signal. At the bottom of Figure 4, we show the calibrated signal produced by the chip. The output eventually converges to a zero-mean, constant-height sine wave independent of the values of the DC offset and amplitude of the input sine wave. Additional experiments have shown that with the input amplitude changing from 20mV to 90mV, the measured output amplitude varies by less than 3mV. Similarly, when the DC offset is varied from 1.5V to 3.5V, the amplitude of the output varies by less than 5mV. These

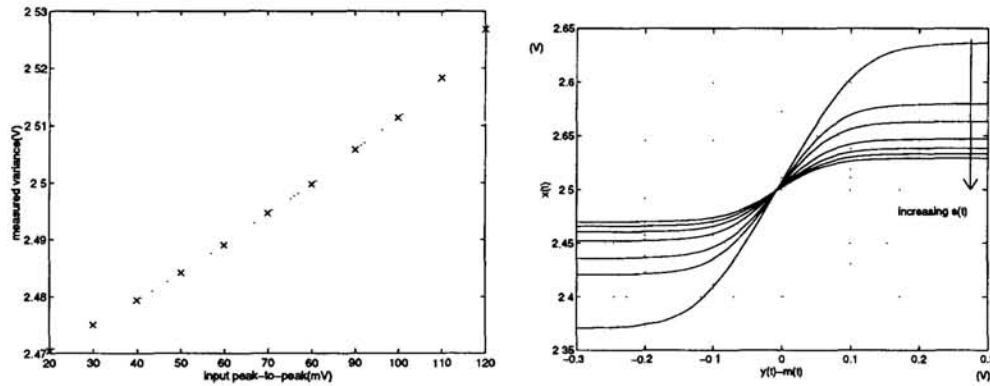

Figure 3: *Left (a) shows the characteristics of measured variance s(t) vs. peak-to-peak input voltage. Right (b) shows the characteristics of divider with different s(t).*

results demonstrate that system is invariant to gain and offset variations of the input.

## 4   Conclusions

The calibration circuit has been demonstrated with the time-constants on the order of 100ms. In many applications, much longer time constants will be necessarily and these cannot be reached with on-chip capacitors even with subthreshold CMOS operation. We expect to use floating-gate techniques where essentially arbitrarily long time-constants can be achieved. Mead has demonstrated a novel adaptive adaptive silicon retina that requires UV light for adaptation to occur [7]. The adaptive silicon retina implemented the constant average brightness constraint. The unoptimized layout area of one of our calibration circuits is about 250x300 um$^2$ in 2um CMOS technology. A future challenge will be to reduce this area and replace the large on-chip capacitors with floating gates.

### Acknowledgments

The authors would like to acknowledge an NSF CAREER Award and Office of Naval Research contract #N00014-94-1-0858.

## Footnotes

[1] For simplicity the signal $s(t)$ will be called the variance estimate throughout the rest of this paper even though technically $s(t)$ is neither the variance nor the standard deviation.

## References

[1] M. Akita, C. Graham, and Y. Hsia. Maintaining an absolute hue in the presence of different background colors. *Vision Research*, 4:539–556, 1964.

[2] V.R. Carlson. Adaptation in the perception of visual velocity. *J. Exp. Psychol.*, 64(2):192–197, 1962.

[3] M.R. Kruer D.A. Scribner and J.M. Killiany. Infrared focal plane array technology. *Proc. IEEE*, 79(1):66–85, 1991.

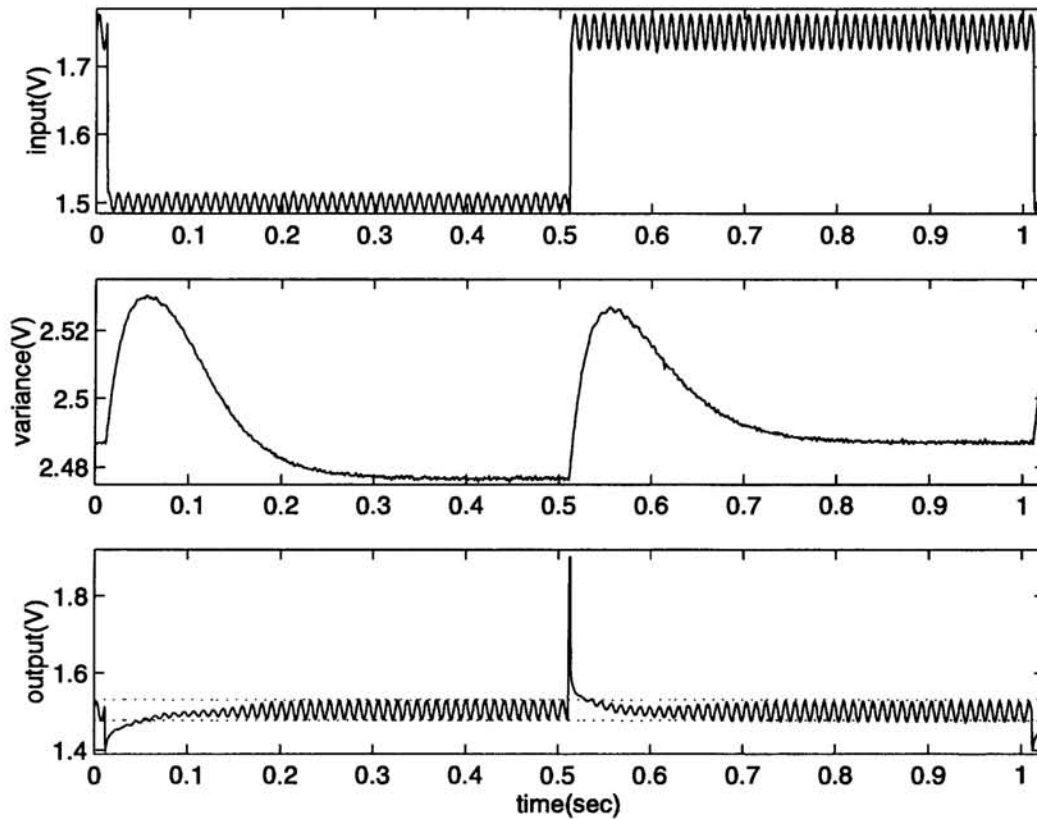

Figure 4: *Calibrating signal with offset and gain variations. Top: the input signal,*
$y(t)$. *Middle: the computed signal variance* $s(t)$. *Bottom: the output signal,* $x(t)$.

[4] T. Delbrück. An electronic photoreceptor sensitive to small changes. In
D. Touretzky, editor, *Advance in Neural Information Processing Systems, Volume 1*, pages 720–727. Morgan Kaufmann, Palo Alto, CA, 1989.

[5] J. Gibson. Adaptation, aftereffect and contrast in the perception of curved
lines. *J. Exp. Psychol.*, 16:1–16, 1933.

[6] J. G. Harris. Continuous-time calibration of VLSI sensors for gain and offset
variations. In *SPIE Internatiional Symposium on Aerospace Sensing and Dual-Use Photonices:Smart Focal Plane Arrays and Focal Plane Array Testing*, pages
23–33, Orlando, FL, April 1995.

[7] C. Mead. Adaptive retina. In C. Mead and M. Ismail, editors, *Analog VLSI Implementation of Neural Systems*, pages 239–246. Kluwer Academic Publishers,
1989.

[8] C. Mead. *Analog VLSI and Neural Systems*. Addison-Wesley, 1989.

[9] D.A. Scribner, K.A. Sarkady, M.R. Kruer, J.T. Calufield, J.D. Hunt, M. Colbert, and M. Descour. Adaptive retina-like preprocessing for imaging detector
arrays. In *Proc. of the IEEE International Conference on Neural Networks*,
pages 1955–1960, San Francisco, CA, Feb. 1993.

[10] S. Ullman and G. Schechtman. Adaptation and gain normalization. *Proc. R.
Soc. Lond. B*, 216:299–313, 1982.
